# Multiple timescales of adaptation in a neural code

**Adrienne L. Fairhall, Geoffrey D. Lewen, William Bialek,**
**and Robert R. de Ruyter van Steveninck**
NEC Research Institute
4 Independence Way
Princeton, New Jersey 08540
*adrienne/geoff/bialek/ruyter@research.nj.nec.com*

## Abstract

Many neural systems extend their dynamic range by adaptation. We examine the timescales of adaptation in the context of dynamically modulated rapidly-varying stimuli, and demonstrate in the fly visual system that adaptation to the statistical ensemble of the stimulus dynamically maximizes information transmission about the time-dependent stimulus. Further, while the rate response has long transients, the adaptation takes place on timescales consistent with optimal variance estimation.

## 1 Introduction

Adaptation was one of the first phenomena discovered when Adrian recorded the responses of single sensory neurons [1, 2]. Since that time, many different forms of adaptation have been found in almost all sensory systems. The simplest forms of adaptation, such as light and dark adaptation in the visual system, seem to involve just discarding a large constant background signal so that the system can maintain sensitivity to small changes. The idea of Attneave [3] and Barlow [4] that the nervous system tries to find an efficient representation of its sensory inputs implies that neural coding strategies should be adapted not just to constant parameters such as the mean light intensity, but to the entire *distribution* of input signals [5]; more generally, efficient strategies for processing (not just coding) of sensory signals must also be matched to the statistics of these signals [6]. Adaptation to statistics might happen on evolutionary time scales, or, at the opposite extreme, it might happen in real time as an animal moves through the world. There is now evidence from several systems for real time adaptation to statistics [7, 8, 9], and at least in one case it has been shown that the form of this adaptation indeed does serve to optimize the efficiency of representation, maximizing the information that a single neuron transmits about its sensory inputs [10].

Perhaps the simplest of statistical adaptation experiments, as in Ref [7] and Fig. 1, is to switch between stimuli that are drawn from different probability distributions and ask how the neuron responds to the switch. When we 'repeat' the experiment we repeat the time dependence of the parameters describing the distribution, but we choose new signals from the same distributions; thus we probe the response or adaptation to the distributions and not to the particular signals. These switching experiments typically reveal transient responses to the switch that have rather long time scales, and it is tempting to identify these long time scales as *the* time scales of adaptation. On the other hand, one can also view the process of adapting to a distribution as one of learning the parameters of that distribution, or of accumulating evidence that the distribution has changed. Some features of the dynamics

in the switching experiments match the dynamics of an optimal statistical estimator [11], but the overall time scale does not: for all the experiments we have seen, the apparent time scales of adaptation in a switching experiment are much longer than would be required to make reliable estimates of the relevant statistical parameters.

In this work we re-examine the phenomena of statistical adaptation in the motion sensitive neurons of the fly visual system. Specifically, we are interested in adaptation to the variance or dynamic range of the velocity distribution [10]. It has been shown that, in steady state, this adaptation includes a rescaling of the neuron's input/output relation, so that the system seems to encode dynamic velocity signals in relative units; this allows the system, presumably, to deal both with the $\sim 50°/\text{s}$ motions that occur in straight flight and with the $\sim 2000°/\text{s}$ motions that occur during acrobatic flight (see Ref.[12]). Further, the precise form of rescaling chosen by the fly's visual system is that which maximizes information transmission. There are several natural questions: (1) How long does it take the system to accomplish the rescaling of its input/output relation? (2) Are the transients seen in switching experiments an indication of gradual rescaling? (3) If the system adapts to the variance of its inputs, is the neural signal ambiguous about the absolute scale of velocity? (4) Can we see the optimization of information transmission occurring in real time?

## 2   Stimulus structure and experimental setup

A fly (*Calliphora vicina*) is immobilized in wax and views a computer controlled oscilloscope display while we record action potentials from the identified neuron H1 using standard methods. The stimulus movie is a random pattern of dark and light vertical bars, and the entire pattern moves along a random trajectory with velocity $S(t)$; since the neuron is motion (and not position) sensitive we refer to this signal as the stimulus. We construct the stimulus $S(t)$ as the product of a normalized white noise $s(t)$, constructed from a random number sequence refreshed every $\tau_s = 2$ ms, and an amplitude or standard deviation $\sigma(t)$ which varies on a characteristic timescale $\tau_a \gg \tau_s$. Frames of the movie are drawn every 2 ms. For analysis all spike times are discretized at the 2 ms resolution of the movie.

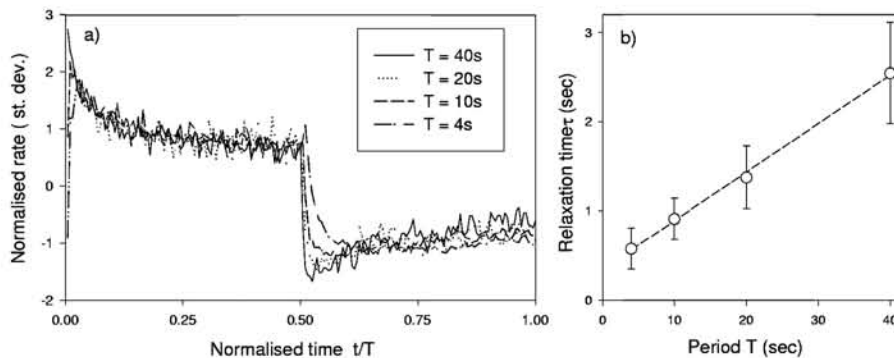

Figure 1: (a) The spike rate measured in response to a square-wave modulated white noise stimulus $s(t)$, averaged over many presentations of $s(t)$, and normalized by the mean and standard deviation. (b) Decay time of the rate following an upward switch as a function of switching period $T$.

## 3   Spike rate dynamics

Switching experiments as described above correspond to a stimulus such that the amplitude $\sigma(t)$ is a square wave, alternating between two values $\sigma_1$ and $\sigma_2$, $\sigma_1 > \sigma_2$. Experiments were performed over a range of switching periods ($T = 40, 20, 10, 4\,\text{s}$), with the amplitudes $\sigma_1$ and $\sigma_2$ in a ratio of 5:1. Remarkably, the timescales of the response depend

strongly on those of the experiment; in fact, the response times rescale by $T$, as is seen in Fig. 1(a). The decay of the rate in the first half of the experiment is fitted by an exponential, and in Fig. 1(b), the resulting decay time $\tau(T)$ is plotted as a function of $T$; we use an exponential not to insist that this is the correct form, only to extract a timescale. As suggested by the rescaling of Fig. 1(a), the fitted decay times are well described as a linear function of the stimulus period. This demonstrates that the timescale of adaptation of the rate is not absolute, but is a function of the timescale established in the experiment.

Large sudden changes in stimulus variance might trigger special mechanisms, so we turn to a signal that changes variance continuously: the amplitude $\sigma(t)$ is taken to be the exponential of a sinusoid, $\sigma(t) = \exp(a\sin(2\pi kt))$, where the period $T = 1/k$ was varied between 2 s and 240 s, and the constant $a$ is fixed such that the amplitude varies by a factor of 10 over a cycle. A typical averaged rate response to the exponential–sinusoidal stimulus is shown in Fig. 2(a). The rate is close to sinusoidal over this parameter regime, indicating a logarithmic encoding of the stimulus variance. Significantly, the rate response shows a phase lead $\Delta\Phi$ with respect to the stimulus. This may be interpreted as the effect of adaptation: at every point on the cycle, the gain of the response is set to a value defined by the stimulus a short time before.

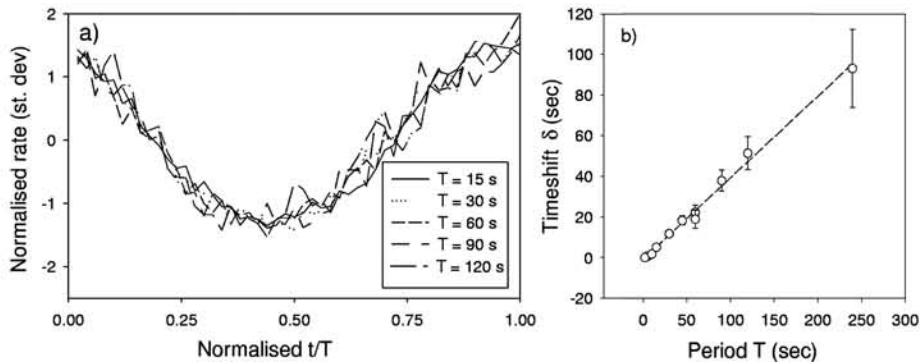

Figure 2: (a) The spike rate measured in response to a exponential-sinusoidal modulation of a white noise stimulus $s(t)$, averaged over presentations of $s(t)$, and normalised by the mean and standard deviation, for several periods T. (b) The time shift $\delta$ between response and stimulus, for a range of periods $T$.

As before, the response of the system was measured over a range of periods $T$. Fig. 2(b) shows the measured relation of the timeshift $\delta(T) = T\Delta\Phi$ of the response as a function of $T$. One observes that the relation is nearly linear over more than one order of magnitude in $T$; that is, the phase shift is approximately constant. Once again there is a strong and simple dependence of the apparent timescale of adaptation on the stimulus parameters. Responses to stimulus sequences composed of many frequencies also exhibit a phase shift, consistent with that observed for the single frequency experiments.

## 4  The dynamic input-output relation

Both the switching and sinusoidally modulated experiments indicate that responses to changing the variance of input signals have multiple time scales, ranging from a few seconds to several minutes. Does it really take the system this long to adjust its input/output relation to the new input distribution? In the range of velocities used, and at the contrast level used in the laboratory, spiking in H1 depends on features of the velocity waveform that occur within a window of $\sim 100$ ms. After a few seconds, then, the system has had access to several tens of independent samples of the motion signal, and should be able to estimate its variance to within $\sim 20\%$; after a minute the precision would be better than a few percent. In practice, we are changing the input variance not by a few percent but a

factor of two or ten; if the system were really efficient, these changes would be detected and compensated by adaptation on much shorter time scales. To address this, we look directly at the input/output relation as the standard deviation $\sigma(t)$ varies in time.

For simplicity we analyze (as in Ref. [10]) features of the stimulus that modulate the probability of occurrence of individual spikes, $P(\text{spike}|\text{stimulus})$; we will not consider patterns of spikes, although the same methods can be easily generalised. The space of stimulus histories of length $\sim 100\,\text{ms}$, discretised at 2 ms, leading up to a spike has a dimensionality $\sim 50$, too large to allow adequate sampling of $P(\text{spike}|\text{stimulus})$ from the data, so we must begin by reducing the dimensionality of the stimulus description.

The simplest way to do so is to find a subset of directions in stimulus space determined to be relevant for the system, and to project the stimulus onto that set of directions. These directions correspond to linear filters. Such a set of directions can be obtained from the moments of the spike-conditional stimulus; the first such moment is the spike-triggered average, or reverse correlation function [2]. It has been shown [10] that for H1, under these conditions, there are two relevant dimensions: a smoothed version of the velocity, and also its derivative. The rescaling observed in steady state experiments was seen to occur independently in both dimensions, so without loss of generality we will use as our filter the single dimension given by the spike-triggered average. The stimulus projected onto this filter will be denoted by $s_0$.

The filtered stimulus is then passed through a nonlinear decision process akin to a threshold. To calculate the input/output relation $P(\text{spike}|s_0)$ [10], we use Bayes' rule:

$$\frac{P(\text{spike}|s_0)}{P(\text{spike})} = \frac{P(s_0|\text{spike})}{P(s_0)}. \tag{1}$$

The spike rate $r(s_0)$ is proportional to the probability of spiking, $r(s_0) \propto P(\text{spike}|s_0)$, leading to the relation

$$\frac{r(s_0)}{\bar{r}} = \frac{P(s_0|\text{spike})}{P(s_0)}, \tag{2}$$

where $\bar{r}$ is the mean spike rate. $P(s_0)$ is the prior distribution of the projected stimulus, which we know. The distribution $P(s_0|\text{spike})$ is estimated from the projected stimulus evaluated at the spike times, and the ratio of the two is the nonlinear input/output relation.

A number of experiments have shown that the filter characteristics of H1 are adaptive, and we see this in the present experiments as well: as the amplitude $\sigma(t)$ is decreased, the filter changes both in overall amplitude and shape. The filter becomes increasingly extended: the system integrates over longer periods of time under conditions of low velocities. Thus the filter depends on the input variance, and we expect that there should be an observable relaxation of the filter to its new steady state form after a switch in variance. We find, however, that within 200 ms following the switch, the amplitude of the filter has already adjusted to the new variance, and further that the detailed shape of the filter has attained its steady state form in less than 1 s. The precise timescale of the establishment of the new filter shape depends on the value of $\sigma$: for the change to $\sigma_1$, the steady state form is achieved within 200 ms. The long tail of the low variance filter for $\sigma_2$ ($< \sigma_1$) is established more slowly. Nonetheless, these time scales which characterize adaptation of the filter are much shorter than the rate transients seen in the switching experiments, and are closer to what we might expect for an efficient estimator.

We construct time dependent input/output relations by forming conditional distributions using spikes from particular time slices in a periodic experiment. In Figs. 3.1(b) and 3.1(c), we show the input/output relation calculated in 1 s bins throughout the switching experiment. Within the first second the input/output relation is almost indistinguishable from its steady state form. Further, it takes the *same form* for the two halves of the experiment: it is rescaled by the standard deviation, as was seen for the steady state experiments. The close collapse or rescaling of the input/output relations depends not only on the normalisation by the standard deviation, but also on the use of the "local" adapted filter (i.e. measured in the same time bin). Returning to the sinusoidal experiments, the input/output relations were

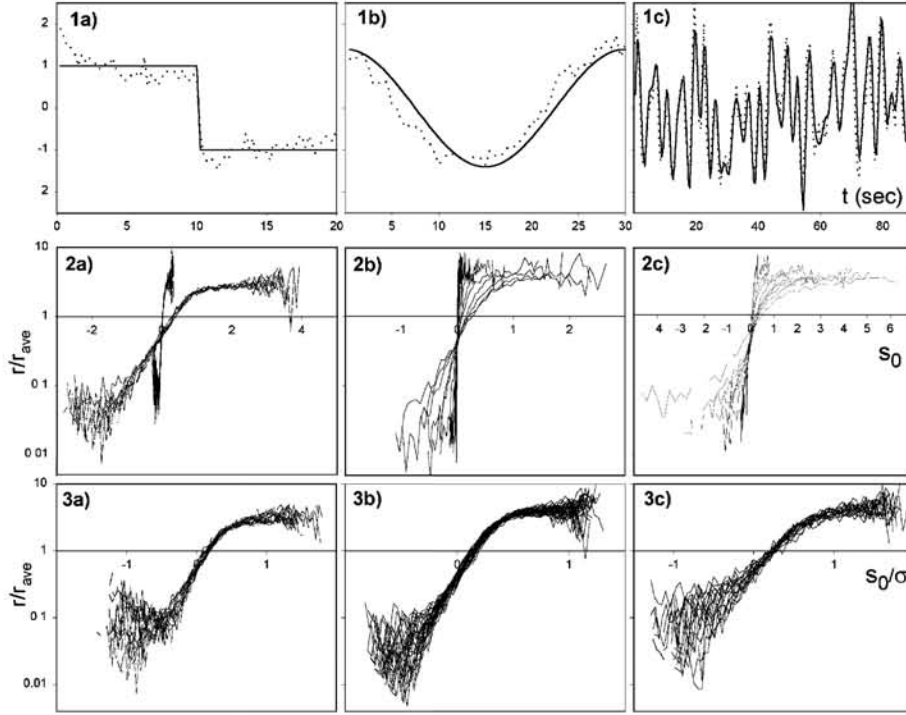

Figure 3: Input/output relations for (a) switching, (b) sinusoidal and (c) randomly modulated experiments. Figs. 3.1 show the modulation envelope $\sigma(t)$, in log for (b) and (c) (solid), and the measured rate (dotted), normalised by mean and standard deviation. Figs. 3.2 show input/output relations calculated in non-overlapping bins throughout the stimulus cycle, with the input $s_0$ in units of the standard deviation of the whole stimulus. Figs. 3.3 show the input/output relations with the input rescaled to units of the local standard deviation.

constructed for $T = 45$ s in 20 non-overlapping bins of width 2.25 s. Once again the functions show a remarkable rescaling which is sharpened by the use of the appropriate local filter: see Fig.3.2(b) and (c). Finally, we consider an amplitude which varies randomly with correlation time $\tau_\sigma \sim 3$ s: $\sigma(t)$ is a repeated segment of the exponential of a Gaussian random process, pictured in Fig.3.3(a), with periodicity $T = 90s \gg \tau_\sigma$. Dividing the stimulus into sequential bins of 2 s in width, we obtain the filters for each timeslice, and calculate the local prior distributions, which are not Gaussian in this case as they are distorted by the local variations of $\sigma(t)$. Nonetheless, the ratio $P(s_0|\text{spike})/P(s_0)$ conspires such that the form of the input/output relation is preserved.

In all three cases, our results show that the system rapidly and continuously adjusts its coding strategy, rescaling the input/output relation with respect to the local variance of the input as for steady state stimuli. Variance normalisation occurs as rapidly as is measurable, and the system chooses a similar form for the input/output relation in each case.

## 5  Information transmission

What does this mean for the coding efficiency of the neuron? An experiment was designed to track the information transmission as a function of time. We use a small set of $N$ 2 s long random noise sequences $\{s_i(t)\}$, $i = 1, \ldots, N$, presented in random order at two

different amplitudes, $\sigma_1$ and $\sigma_2$. We then ask how much information the spike train conveys about (a) which of the random segments $s_i(t)$ and (b) which of the amplitudes $\sigma_j$ was used. Specifically, the experiment consists of a series of trials of length 2 s where the fast component is one of the sequences $\{s_i\}$, and after 1 s, the amplitude switches from $\sigma_1$ to $\sigma_2$ or vice versa. $N$ was taken to be 40, so that a 2 hour experiment provides approximately 80 samples for each $(s_i, \sigma_j)$. This allows us to measure the mutual information between the response and either the fast or the slow component of the stimulus as a function of time across the 2 s repeated segment. We use only a very restricted subspace of $\sigma$ and $s$: the maximum available information about $\sigma$ is 1 bit, and about $s$ is $\log_2 N$.

The spike response is represented by "words" [13], generated from the spike times discretised to timebins of 2 ms, where no spike is represented by 0, and a spike by 1. A word is defined as the binary digit formed from 10 consecutive bins, so there are $2^{10}$ possible words. The information about the fast component $s$ in trials of a given $\sigma$ is

$$I_\sigma(w(t); s) = H[P_\sigma(w(t))] - \sum_{j=1}^{N} P(s_j) H[P_\sigma(w(t); s_j)], \tag{3}$$

where $H$ is the entropy of the word distribution:

$$H[P(w(t))] = -\sum_k P(w_k(t)) \log_2 P(w_k(t)). \tag{4}$$

One can compare this information for different values of $\sigma$. Similarly, one can calculate the information about the amplitude using a given probe $s$:

$$I_s(w(t); \sigma) = H[P_s(w(t))] - \sum_{j=1}^{2} P(\sigma_j) H[P_s(w(t); \sigma_j)]. \tag{5}$$

The amount of information for each $s_j$ varies rapidly depending on the presence or absence of spikes, so we average these contributions over the $\{s_j\}$ to give $I(w; \sigma)$.

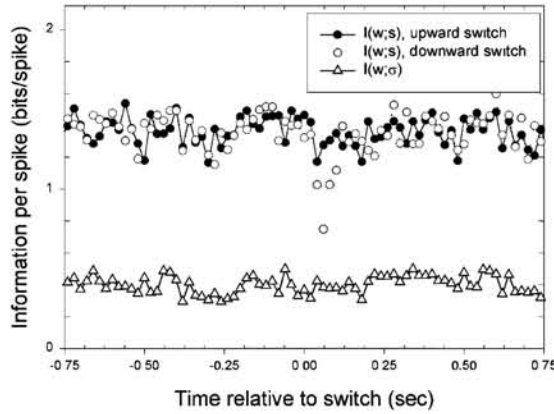

Figure 4: Information per spike as a function of time where $\sigma$ is switched every 2 s.

The mutual information as a function of time is shown in Fig. 4, presented as bits/spike. As one would expect, the amount of information transmitted *per second* about the stimulus details, or $s$, depends on the ensemble parameter $\sigma$: larger velocities allow a higher SNR for velocity estimation, and the system is able to transmit more information. However,

when we convert the information rate to bits/spike, we find that the system is transmitting at a constant efficiency of around 1.4 bits/spike. Any change in information rate during a switch from $\sigma_1$ to $\sigma_2$ is undetectable. For a switch from $\sigma_2$ to $\sigma_1$, the time to recovery is of order 100 ms. This demonstrates explicitly that the system is indeed rapidly maximising its information transmission. Further, the transient "excess" of spikes following an upward switch provide information at a constant rate per spike. The information about the amplitude, similarly, remains at a constant level throughout the trial. Thus, information about the ensemble variable is retained at all times: the response is *not ambiguous* with respect to the absolute scale of velocity. Despite the rescaling of input/output curves, responses within different ensembles are distinguishable.

## 6   Discussion

We find that the neural response to a stimulus with well-separated timescales $S(t) = \sigma(t)s(t)$ takes the form of a **rate⊗timing** code, where the response $r(t)$ may be approximately modelled as

$$r(t) = R[\sigma(t)]g\left(s(t)\right). \qquad (6)$$

Here $R$ modulates the overall rate and depends on the slow dynamics of the variance envelope, while the precise timing of a given spike in response to fast events in the stimulus is determined by the nonlinear input/output relation $g$, which depends only on the normalised quantity $s(t)$. Through this apparent normalisation by the local standard deviation, $g$, as for steady-state experiments, maximises information transmission about the fast components of the stimulus. The function $R$ modulating the rate varies on much slower timescales so cannot be taken as an indicator of the extent of the system's adaptation to a new ensemble. Rather, $R$ appears to function as an independent degree of freedom, capable of transmitting information, at a slower rate, about the slow stimulus modulations. The presence of many timescales in $R$ may itself be an adaptation to the many timescales of variation in natural signals. At the same time, the rapid readjustment of the input/output relation – and the consequent recovery of information after a sudden change in $\sigma$ – indicate that the adaptive mechanisms approach the limiting speed set by the need to gather statistics.

**Acknowledgments**

We thank B. Agüera y Arcas, N. Brenner and T. Adelman for helpful discussions.

**References**

[1]  E. Adrian (1928) The Basis of Sensation (London Christophers)

[2]  F. Rieke, D. Warland, R. de Ruyter van Steveninck and W. Bialek (1997). Spikes: exploring the neural code. (Cambridge, MA: MIT Press).

[3]  F. Attneave (1954) *Psych. Rev. 61*, 183-193.

[4]  H. B. Barlow (1961) in Sensory Communication, W. A. Rosenbluth, ed. (Cambridge, MA: MIT Press), pp.217-234.

[5]  S.B. Laughlin (1981) *Z. Naturforsch. 36c*, 910-912.

[6]  M. Potters and W. Bialek (1994) *J. Phys. I. France 4*, 1755-1775.

[7]  S. Smirnakis, M. Berry, D. Warland, W. Bialek and M. Meister (1997) *Nature* 386, 69-73.

[8]  J. H. van Hateren (1997) *Vision Research 37*, 3407-3416.

[9]  R.R. de Ruyter van Steveninck, W. Bialek, M. Potters, and R. H. Carlson (1994) Proc. of the IEEE International Conference on Systems, Man and Cybernetics, 302-307.

[10]  N. Brenner, W. Bialek and R. de Ruyter van Steveninck (2000), *Neuron, 26*, 695-702.

[11]  M. deWeese and A. Zador (1998) *Neural Comp. 10*, 1179-1202.

[12]  C. Schilstra and J. H. van Hateren (1999) *J. Exp. Biol. 202*, 1481-1490.

[13]  S. Strong, R. Koberle, R. de Ruyter van Steveninck and W. Bialek (1998) *Phys. Rev. Lett. 80*, 197-200.